# Ultrafast Monte Carlo for Kernel Estimators and Generalized Statistical Summations

**Michael P. Holmes, Alexander G. Gray, and Charles Lee Isbell, Jr.**
College Of Computing
Georgia Institute of Technology
Atlanta, GA 30327
{mph, agray, isbell}@cc.gatech.edu

## Abstract

Machine learning contains many computational bottlenecks in the form of nested summations over datasets. Kernel estimators and other methods are burdened by these expensive computations. Exact evaluation is typically $O(n^2)$ or higher, which severely limits application to large datasets. We present a multi-stage stratified Monte Carlo method for approximating such summations with probabilistic relative error control. The essential idea is fast approximation by sampling in trees. This method differs from many previous scalability techniques (such as standard multi-tree methods) in that its error is stochastic, but we derive conditions for error control and demonstrate that they work. Further, we give a theoretical sample complexity for the method that is independent of dataset size, and show that this appears to hold in experiments, where speedups reach as high as $10^{14}$, many orders of magnitude beyond the previous state of the art.

## 1 Introduction

Many machine learning methods have computational bottlenecks in the form of nested summations that become intractable for large datasets. We are particularly motivated by the nonparametric kernel estimators (e.g. kernel density estimation), but a variety of other methods require computations of similar form. In this work we formalize the general class of nested summations and present a new multi-stage Monte Carlo method for approximating any problem in the class with rigorous relative error control. Key to the efficiency of this method is the use of tree-based data stratification, i.e. sampling in trees. We derive error guarantees and sample complexity bounds, with the intriguing result that runtime depends not on dataset size but on statistical features such as variance and kurtosis, which can be controlled through stratification. We also present experiments that validate these theoretical results and demonstrate tremendous speedup over the prior state of the art.

Previous approaches to algorithmic acceleration of this kind fall into roughly two groups: 1) methods that run non-accelerated algorithms on subsets of the data, typically without error bounds, and 2) multi-tree methods with deterministic error bounds. The former are of less interest due to the lack of error control, while the latter are good when exact error control is required, but have built-in overconservatism that limits speedup, and are difficult to extend to new problems. Our Monte Carlo approach offers much larger speedup and a generality that makes it simple to adapt to new problems, while retaining strong error control. While there are non-summative problems to which the standard multi-tree methodology is applicable and our Monte Carlo method is not, our method appears to give greater speedup by many orders of magnitude on problems where both methods can be used.

In summary, this work makes the following contributions: formulation of the class of generalized nested data summations; derivation of recursive Monte Carlo algorithms with rigorous error guarantees for this class of computation; derivation of sample complexity bounds showing no explicit

dependence on dataset size; variance-driven tree-based stratified sampling of datasets, which allows Monte Carlo approximation to be effective with small sample sizes; application to kernel regression and kernel conditional density estimation; empirical demonstration of speedups as high as $10^{14}$ on datasets with points numbering in the millions. It is the combination of all these elements that enables our method to perform so far beyond the previous state of the art.

## 2 Problem definition and previous work

We first illustrate the problem class by giving expressions for the least-squares cross-validation scores used to optimize bandwidths in kernel regression (KR), kernel density estimation (KDE), and kernel conditional density estimation (KCDE):

$$S_{KR} = \frac{1}{n} \sum_i \left( y_i - \frac{\sum_{j \neq i} K_h(||x_i - x_j||) y_j}{\sum_{j \neq i} K_h(||x_i - x_j||)} \right)^2$$

$$S_{KDE} = \frac{1}{n} \sum_i \left( \frac{1}{(n-1)^2} \sum_{j \neq i} \sum_{k \neq i} \int K_h(||x - x_j||) K_h(||x - x_k||) dx - \frac{2}{(n-1)} \sum_{j \neq i} K_h(||x_i - x_j||) \right)$$

$$S_{KCDE} = \frac{1}{n} \sum_i \left( \frac{\sum_{j \neq i} \sum_{k \neq i} K_{h_2}(||x_i - x_j||) K_{h_2}(||x_i - x_k||) \int K_{h_1}(y - y_j) K_{h_1}(y - y_k) dy}{\sum_{j \neq i} \sum_{k \neq i} K_{h_2}(||x_i - x_j||) K_{h_2}(||x_i - x_k||)} \right.$$
$$\left. - 2 \frac{\sum_{j \neq i} K_{h_2}(||x_i - x_j||) K_{h_1}(y_i - y_j)}{\sum_{j \neq i} K_{h_2}(||x_i - x_j||)} \right).$$

These nested sums have quadratic and cubic computation times that are intractable for large datasets. We would like a method for quickly approximating these and similar computations in a simple and general way. We begin by formulating an inductive generalization of the problem class:

$$B(\mathcal{X}_c) \to \sum_{i \in I(\mathcal{X}_c)} f(\mathcal{X}_c, \mathcal{X}_i) \tag{1}$$

$$G(\mathcal{X}_c) \to B(\mathcal{X}_c) \; | \; \sum_{i \in I(\mathcal{X}_c)} f(\mathcal{X}_c, G_1(\mathcal{X}_c, \mathcal{X}_i), G_2(\mathcal{X}_c, \mathcal{X}_i), \dots). \tag{2}$$

$B$ represents the base case, in which a tuple of constant arguments $\mathcal{X}_c$ may be specified and a tuple of variable arguments $\mathcal{X}_i$ is indexed by a set $I$, which may be a function of $\mathcal{X}_c$. For instance, in the innermost leave-one-out summations of $S_{KR}$, $\mathcal{X}_c$ is the single point $x_i$ while $I(\mathcal{X}_c)$ indexes all single points other than $x_i$. Note that $|I|$ is the number of terms in a summation of type $B$, and therefore represents the base time complexity. Whenever $I$ consists of all $k$-tuples or leave-one-out $k$-tuples, the base complexity is $O(n^k)$, where $n$ is the size of the dataset.

The inductive case $G$ is either: 1) the base case $B$, or 2) a sum where the arguments to the summand function are $\mathcal{X}_c$ and a series of nested instances of type $G$. In $S_{KR}$ the outermost summation is an example of this. The base complexity here is $|I|$ multiplied by the maximum base complexity among the nested instances, e.g. if, as in $S_{KR}$, $I$ is all single points and the most expensive inner $G$ is $O(n)$, then the overall base complexity is $O(n^2)$.

**Previous work.** Past efforts at scaling this class of computation have fallen into roughly two groups. First are methods where data is simply subsampled before running a non-accelerated algorithm. Stochastic gradient descent and its variants (e.g. [1]) are prototypical here. While these approaches can have asymptotic convergence, there are no error guarantees for finite sample sizes. This is not show-stopping in practice, but the lack of quality assurance is a critical shortcoming. Our approach also exploits the speedup that comes from sampling, but provides a rigorous relative error guarantee and is able to automatically determine the necessary sample size to provide that guarantee.

The other main class of acceleration methods consists of those employing "higher order divide and conquer" or multi-tree techniques that give either exact answers or deterministic error bounds (e.g. [2, 3, 4]). These approaches apply to a broad class of "generalized n-body problems" (GNPs), and feature the use of multiple spatial partitioning structures such as $kd$-trees or ball trees to decompose and reuse portions of computational work. While the class of GNPs has yet to be formally defined, the generalized summations we address are clearly related and have at least partial overlap.

The standard multi-tree methodology has three significant drawbacks. First, although it gives deterministic error bounds, the bounds are usually quite loose, resulting in overconservatism that prevents aggressive approximation that could give greater speed. Second, creating a new multi-tree method to accelerate a given algorithm requires complex custom derivation of error bounds and pruning rules. Third, the standard multi-tree approach is conjectured to reduce $O(n^p)$ computations at best to $O(n^{\log p})$. This still leaves an intractable computation for $p$ as small as 4.

In [5], the first of these concerns began to be addressed by employing sample-based bounds within a multi-tree error propagation framework. The present work builds on that idea by moving to a fully Monte Carlo scheme where multiple trees are used for variance-reducing stratification. Error is rigorously controlled and driven by sample variance, allowing the Monte Carlo approach to make aggressive approximations and avoid the overconservatism of deterministic multi-tree methods. This yields greater speedups by many orders of magnitude. Further, our Monte Carlo approach handles the class of nested summations in full generality, making it easy to specialize to new problems. Lastly, the computational complexity of our method is not directly dependent on dataset size, which means it can address high degrees of nesting that would make the standard multi-tree approach intractable. The main tradeoff is that Monte Carlo error bounds are probabilistic, though the bound probability is a parameter to the algorithm. Thus, we believe the Monte Carlo approach is superior for all situations that can tolerate minor stochasticity in the approximated output.

## 3   Single-stage Monte Carlo

We first derive a Monte Carlo approximation for the base case of a single-stage, flat summation, i.e. Equation 1. The basic results for this simple case (up to and including Algorithm 1 and Theorem 1) mirror the standard development of Monte Carlo as in [6] or [7], with some modification to accommodate our particular problem setup. We then move beyond to present novel sample complexity bounds and extend the single-stage results to the multi-stage and multi-stage stratified cases. These extensions allow us to efficiently bring Monte Carlo principles to bear on the entire class of generalized summations, while yielding insights into the dependence of computational complexity on sample statistics and how tree-based methods can improve those statistics.

To begin, note that the summation $B\left(\mathcal{X}_c\right)$ can be written as $nE[f_i] = n\mu_f$, where $n = |I|$ and the expectation is taken over a discrete distribution $P_f$ that puts mass $\frac{1}{n}$ on each term $f_i = f(\mathcal{X}_c, \mathcal{X}_i)$. Our goal is to produce an estimate $\hat{B}$ that has low relative error with high probability. More precisely, for a specified $\epsilon$ and $\alpha$, we want $|\hat{B} - B| \leq \epsilon|B|$ with probability at least $1 - \alpha$. This is equivalent to estimating $\mu_f$ by $\hat{\mu}_f$ such that $|\hat{\mu}_f - \mu_f| \leq \epsilon|\mu_f|$. Let $\hat{\mu}_f$ be the sample mean of $m$ samples taken from $P_f$. From the Central Limit Theorem, we have asymptotically $\hat{\mu}_f \rightsquigarrow N(\mu_f, \hat{\sigma}_f^2/m)$, where $\hat{\sigma}_f^2$ is the sample variance, from which we can construct the standard confidence interval: $|\hat{\mu}_f - \mu_f| \leq z_{\alpha/2}\hat{\sigma}_f/\sqrt{m}$ with probability $1 - \alpha$. When $\hat{\mu}_f$ satisfies this bound, our relative error condition is implied by $z_{\alpha/2}\hat{\sigma}_f/\sqrt{m} \leq \epsilon|\mu_f|$, and we also have $|\mu_f| \geq |\hat{\mu}_f| - z_{\alpha/2}\hat{\sigma}_f/\sqrt{m}$. Combining these, we can ensure our target relative error by requiring that $z_{\alpha/2}\hat{\sigma}_f/\sqrt{m} \leq \epsilon(|\hat{\mu}_f| - z_{\alpha/2}\hat{\sigma}_f/\sqrt{m})$, which rearranges to:

$$m \geq z_{\alpha/2}^2 \frac{(1+\epsilon)^2}{\epsilon^2} \frac{\hat{\sigma}_f^2}{\hat{\mu}_f^2} \ . \tag{3}$$

Equation 3 gives an empirically testable condition that guarantees the target relative error level with probability $1 - \alpha$, given that $\hat{\mu}_f$ has reached its asymptotic distribution $N(\mu_f, \hat{\sigma}_f^2/m)$. This suggests an iterative sampling procedure in which $m$ starts at a value $m_{min}$ chosen to make the normal approximation valid, and then is increased until the condition of Equation 3 is met. This procedure is summarized in Algorithm 1, and we state its error guarantee as a theorem.

**Theorem 1.** *Given $m_{min}$ large enough to put $\hat{\mu}_f$ in its asymptotic normal regime, with probability at least $1 - \alpha$ Algorithm 1 approximates the summation S with relative error no greater than $\epsilon$ .*

*Proof.* We have already established that Equation 3 is a sufficient condition for $\epsilon$ relative error with probability $1 - \alpha$. Algorithm 1 simply increases the sample size until this condition is met. ☐

**Sample Complexity.** Because we are interested in *fast* approximations, Algorithm 1 is only useful if it terminates with $m$ significantly smaller than the number of terms in the full summation. Equation 3

---

**Algorithm 1** Iterative Monte Carlo approximation for flat summations.

---

**MC-Approx**($S$, $\mathcal{X}_c$, $\epsilon$, $\alpha$, $m_{min}$)
  $samples \leftarrow \emptyset$, $m_{needed} \leftarrow m_{min}$
  **repeat**
    $addSamples(samples, m_{needed}, S, \mathcal{X}_c)$
    $m$, $\hat{\mu}_f$, $\hat{\sigma}_f^2 \leftarrow calcStats(samples)$
    $m_{thresh} \leftarrow z_{\alpha/2}^2 (1+\epsilon)^2 \hat{\sigma}_f^2 / \epsilon^2 \hat{\mu}_f^2$
    $m_{needed} \leftarrow m_{thresh} - m$
  **until** $m \geq m_{thresh}$
  **return** $|S.I|\hat{\mu}_f$

**addSamples**($samples$, $m_{needed}$, $S$, $\mathcal{X}_c$)
  **for** $i = 1$ to $m_{needed}$
    $\mathcal{X}_i \leftarrow rand(S.I)$
    $samples \leftarrow samples \cup S.f(\mathcal{X}_c, \mathcal{X}_i)$
  **end for**

**calcStats**($samples$)
  $m \leftarrow count(samples)$
  $\hat{\mu}_f \leftarrow avg(samples)$
  $\hat{\sigma}_f^2 \leftarrow var(samples)$
  **return** $m$, $\hat{\mu}_f$, $\hat{\sigma}_f^2$

---

gives an empirical test indicating when $m$ is large enough for sampling to terminate; we now provide an upper bound, in terms of the distributional properties of the full set of $f_i$, for the value of $m$ at which Equation 3 will be satisfied.

**Theorem 2.** *Given $m_{min}$ large enough to put $\hat{\mu}_f$ and $\hat{\sigma}_f$ in their asymptotic normal regimes, with probability at least $1 - 2\alpha$ Algorithm 1 terminates with $m \leq O\big(\frac{\sigma_f^2}{\mu_f^2} + \frac{\sigma_f}{|\mu_f|}\sqrt{\frac{\mu_{4f}}{\sigma_f^4} - 1}\big)$.*

*Proof.* The termination condition is driven by $\hat{\sigma}_f^2/\hat{\mu}_f^2$, so we proceed by bounding this ratio. First, with probability $1 - \alpha$ we have a lower bound on the absolute value of the sample mean: $|\hat{\mu}_f| \geq |\mu_f| - z_{\alpha/2}\hat{\sigma}_f/\sqrt{m}$. Next, because the sample variance is asymptotically distributed as $N(\sigma_f^2, (\mu_{4f} - \sigma_f^4)/m)$, where $\mu_{4f}$ is the fourth central moment, we can apply the delta method to infer that $\hat{\sigma}_f$ converges in distribution to $N(\sigma_f, (\mu_{4f} - \sigma_f^4)/4\sigma_f^2 m)$. Using the normal-based confidence interval, this gives the following $1 - \alpha$ upper bound for the sample standard deviation: $\hat{\sigma}_f \leq \sigma_f + z_{\alpha/2}\sqrt{\mu_{4f} - \sigma_f^4}/(2\sigma_f\sqrt{m})$. We now combine these bounds, but since we only know that each bound individually covers at least a $1 - \alpha$ fraction of outcomes, we can only guarantee they will jointly hold with probability at least $1 - 2\alpha$, giving the following $1 - 2\alpha$ bound:

$$\frac{\hat{\sigma}_f}{|\hat{\mu}_f|} \leq \frac{\sigma_f + z_{\alpha/2}\frac{\sqrt{\mu_{4f}-\sigma_f^4}}{2\sigma_f\sqrt{m}}}{|\mu_f| - z_{\alpha/2}\frac{\sigma_f}{\sqrt{m}}} \ .$$

Combining this with Equation 3 and solving for $m$ shows that, with probability at least $1 - 2\alpha$, the algorithm will terminate with $m$ no larger than:

$$\frac{z_{\alpha/2}^2}{2}\frac{(1+2\epsilon)^2}{\epsilon^2}\frac{\sigma_f}{|\mu_f|}\left[\frac{\sigma_f}{|\mu_f|} + \frac{\epsilon(1+\epsilon)}{(1+2\epsilon)^2}\sqrt{\frac{\mu_{4f}}{\sigma_f^4}-1} + \sqrt{\frac{\sigma_f}{|\mu_f|}}\sqrt{\frac{\sigma_f}{|\mu_f|} + \frac{2\epsilon(1+\epsilon)}{(1+2\epsilon)^2}\sqrt{\frac{\mu_{4f}}{\sigma_f^4}-1}}\right] \ . \quad (4)$$

□

Three aspects of this bound are salient. First, computation time is liberated from dataset size. This is because the sample complexity depends only on the *distributional features* ($\sigma_f^2$, $\mu_f$, and $\mu_{4f}$) of the summation terms, and not on the *number* of terms. For i.i.d. datasets in particular, these distributional features are convergent, which means the sample or computational complexity converges to a constant while speedup becomes unbounded as the dataset size goes to infinity.

Second, the bound has sensible dependence on $\sigma_f/|\mu_f|$ and $\mu_{4f}/\sigma_f^4$. The former is a standard dispersion measure known as the coefficient of variation, and the latter is the kurtosis. Algorithm 1 therefore gives greatest speedup for summations whose terms have low dispersion and low kurtosis. The intuition is that sampling is most efficient when values are concentrated tightly in a few clusters, making it easy to get a representative sample set. This motivates the additional speedup we later gain by stratifying the dataset into low-variance regions.

Finally, the sample complexity bound indicates whether Algorithm 1 will actually give speedup for any particular problem. For a given summation, let the speedup be defined as the total number of terms $n$ divided by the number of terms evaluated by the approximation. For a desired speedup $\tau$, we need $n \geq \tau m_{bound}$, where $m_{bound}$ is the expression in Equation 4. This is the fundamental characterization of whether speedup will be attained.

---

**Algorithm 2** Iterative Monte Carlo approximation for nested summations.

---

**MC-Approx**: as in Algorithm 1  
**calcStats**: as in Algorithm 1

**addSamples**($samples$, $m_{needed}$, $S$, $\mathcal{X}_c$)  
  **for** $i = 1$ to $m_{needed}$  
    $\mathcal{X}_i \leftarrow rand(S.I(\mathcal{X}_c))$  
    $mcArgs \leftarrow map(\textbf{MC-Approx}(*, \ \mathcal{X}_c \circ \mathcal{X}_i, \ \ldots), \ \langle S.G_j \rangle)$  
    $samples \leftarrow samples \ \cup \ S.f(\mathcal{X}_c, \ mcArgs)$  
  **end for**

---

## 4 Multi-stage Monte Carlo

We now turn to the inductive case of nested summations, i.e. Equation 2. The approach we take is to apply the single-stage Monte Carlo algorithm over the terms $f_i$ as before, but with recursive invocation to obtain approximations for the arguments $G_j$. Algorithm 2 specifies this procedure.

**Theorem 3.** *Given $m_{min}$ large enough to put $\hat{\mu}_f$ in its asymptotic normal regime, with probability at least $1 - \alpha$ Algorithm 2 approximates the summation S with relative error no greater than $\epsilon$ .*

*Proof.* We begin by noting that the proof of correctness for Algorithm 1 rests on 1) the ability to sample from a distribution $P_f$ whose expectation is $\mu_f = \frac{1}{n} \sum_i f_i$, and 2) the ability to invoke the CLT on the sample mean $\hat{\mu}_f$ in terms of the sample variance $\hat{\sigma}_f^2$. Given these properties, Equation 3 follows as a sufficient condition for relative error no greater than $\epsilon$ with probability at least $1 - \alpha$. We therefore need only establish that Algorithm 2 samples from a distribution having these properties.

For each sampled $f_i$, let $\widehat{G}_j$ be the recursive approximation for argument $G_j$. We assume $\widehat{G}_j$ has been drawn from a CLT-type normal distribution. Because the $\widehat{G}_j$ are recursively approximated, this is an inductive hypothesis, with the remainder of the proof showing that if the hypothesis holds for the recursive invocations, it also holds for the outer invocation. The base case, where all recursions must bottom out, is the type-$B$ summation already shown to give CLT-governed answers (see proof of Theorem 1). Let $\widehat{\mathcal{G}}_m = (\widehat{G}_1, \widehat{G}_2, \ldots)$ be the vector of $\widehat{G}_j$ values after each $\widehat{G}_j$ has been estimated from $m_j$ samples ($\sum m_j = m$), and let $\mathcal{G}$ be the vector of true $G_j$ values. Since each component $\widehat{G}_j$ converges in distribution to $N(G_j, \sigma_j^2/m_j)$, $\widehat{\mathcal{G}}_m$ satisfies $\widehat{\mathcal{G}}_m \rightsquigarrow N(\mathcal{G}, \Sigma_m)$. We leave the detailed entries of the covariance $\Sigma_m$ unspecified, except to note that its $jj$th element is $\sigma_j^2/m_j$, and that its off-diagonal elements may be non-zero if the $\widehat{G}_j$ are generated in a correlated way (this can be used as a variance reduction technique).

Given the asymptotic normality of $\widehat{\mathcal{G}}_m$, the same arguments used to derive the multivariate delta method can be used, with some modification, to show that $f_i(\widehat{\mathcal{G}}_m) \rightsquigarrow N(f_i(\mathcal{G}), \nabla_f(\mathcal{G})\Sigma_m \nabla_f^T(\mathcal{G}))$. Thus, asymptotically, $f_i(\widehat{\mathcal{G}}_m)$ is normally distributed around its true value with a variance that depends on both the gradient of $f$ and the covariance matrix of the approximated arguments in $\widehat{\mathcal{G}}_m$.

This being the case, uniform sampling of the recursively estimated $f_i$ is equivalent to sampling from a distribution $\tilde{P}_f$ that gives weight $\frac{1}{n}$ to a normal distribution centered on each $f_i$. The expectation over $\tilde{P}_f$ is $\mu_f$, and since the algorithm uses a simple sample mean the CLT does apply. These are the properties we need for correctness, and the applicability of the CLT combined with the proven base case completes the inductive proof. □

Note that the variance over $\tilde{P}_f$ works out to $\tilde{\sigma}_f^2 = \sigma_f^2 + \frac{1}{n} \sum_{i \in I} \sigma_i^2$, where $\sigma_i^2 = \nabla_f(\mathcal{G})\Sigma_m \nabla_f^T(\mathcal{G})$. In other words, the variance with recursive approximation is the exact variance $\sigma_f^2$ plus the average of the variances $\sigma_i^2$ of the approximated $f_i$. Likewise one could write an expression for the kurtosis $\tilde{\mu}_{4f}$. Because we are still dealing with a sample mean, Theorem 2 still holds in the nested case.

**Corollary 2.1.** *Given $m_{min}$ large enough to put $\hat{\mu}_f$ and $\hat{\sigma}_f$ in their asymptotic normal regimes, with probability at least $1 - 2\alpha$ Algorithm 2 terminates with $m \leq O\big(\frac{\tilde{\sigma}_f^2}{\mu_f^2} + \frac{\tilde{\sigma}_f}{|\mu_f|}\sqrt{\frac{\tilde{\mu}_{4f}}{\tilde{\sigma}_f^4} - 1}\big)$.*

It is important to point out that the $1 - \alpha$ confidences and $\epsilon$ relative error bounds of the recursively approximated arguments do *not* pass through to or compound in the overall estimator $\hat{\mu}_f$: their influence appears in the variance $\sigma_i^2$ of each sampled $f_i$, which in turn contributes to the overall variance $\tilde{\sigma}_f^2$, and the error from $\tilde{\sigma}_f^2$ is independently controlled by the outermost sampling procedure.

**Algorithm 3** Iterative Monte Carlo approximation for nested summations with stratification.

**MC-Approx**: as in Algorithm 1

**calcStats**($strata,\ samples$)
  $m \leftarrow count(samples)$
  $\hat{\mu}_{fs} \leftarrow stratAvg(strata, samples)$
  $\hat{\sigma}^2_{fs} \leftarrow stratVar(strata, samples)$
  **return** $m,\ \hat{\mu}_{fs},\ \hat{\sigma}^2_{fs}$

**addSamples**($strata,\ samples,\ m_{needed},\ S,\ \mathcal{X}_c$)
  $needPerStrat = optAlloc(samples, strata, m_{needed})$
  **for** $s = 1$ to $strata.count$
    $m_s = needPerStrat[s]$
    **for** $i = 1$ to $m_s$
      $\mathcal{X}_i \leftarrow rand(S.I(\mathcal{X}_c),\ strata[s])$
      $mcArgs \leftarrow map(\textbf{MC-Approx}(*,\ \mathcal{X}_c \circ \mathcal{X}_i,\ \ldots),\ \langle S.G_j \rangle)$
      $samples[s] \leftarrow samples[s]\ \cup\ S.f(\mathcal{X}_c,\ mcArgs)$
    **end for**
  **end for**

## 5 Variance Reduction

With Algorithm 2 we have coverage of the entire generalized summation problem class, and our focus turns to maximizing efficiency. As noted above, Theorem 2 implies we need fewer samples when the summation terms are tightly concentrated in a few clusters. We formalize this by spatially partitioning the data to enable a stratified sampling scheme. Additionally, by use of correlated sampling we induce covariance between recursively estimated summations whenever the overall variance can be reduced by doing so. Adding these techniques to recursive Monte Carlo makes for an extremely fast, accurate, and general approximation scheme.

**Stratification.** Stratification is a standard Monte Carlo principle whereby the values being sampled are partitioned into subsets (strata) whose contributions are separately estimated and then combined. The idea is that strata with higher variance can be sampled more heavily than those with lower variance, thereby making more efficient use of samples than in uniform sampling. Application of this principle requires the development of an effective partitioning scheme for each new domain of interest. In the case of generalized summations, the values being sampled are the $f_i$, which are not known *a priori* and cannot be directly stratified. However, since $f$ is generally a function with some degree of continuity, its output is similar for similar values of its arguments. We therefore stratify the argument space, i.e. the input datasets, by use of spatial partitioning structures. Though any spatial partitioning could be used, in this work we use modified $kd$-trees that recursively split the data along the dimension of highest variance. The approximation procedure runs as it did before, except that the sampling and sample statistics are modified to make use of the trees. Trees are expanded up to a user-specified number of nodes, prioritized by a heuristic of expanding nodes in order of largest size times average per-dimensional standard deviation. This heuristic will later be justified by the variance expression for the stratified sample mean. The approximation procedure is summarized in Algorithm 3, and we now establish its error guarantee.

**Theorem 4.** *Given $m_{min}$ large enough to put $\hat{\mu}_f$ in its asymptotic normal regime, with probability at least $1 - \alpha$ Algorithm 3 approximates the summation S with relative error no greater than $\epsilon$ .*

*Proof.* Identical to Theorem 3, but we need to establish that 1) the sample mean remains unbiased under stratification, and 2) the CLT still holds under stratification. These turn out to be standard properties of the stratified sample mean and its variance estimator (see [7]):

$$\hat{\mu}_{fs} = \sum_j p_j \hat{\mu}_j \qquad (5)$$

$$\hat{\sigma}^2(\hat{\mu}_{fs}) = \frac{\hat{\sigma}^2_{fs}}{m}; \quad \hat{\sigma}^2_{fs} \triangleq m \sum_j p_j^2 \frac{\hat{\sigma}^2_j}{m_j} = \sum_j \frac{p_j^2}{q_j} \hat{\sigma}^2_j\ , \qquad (6)$$

where $j$ indexes the strata, $\hat{\mu}_j$ and $\hat{\sigma}^2_j$ are the sample mean and variance of stratum $j$, $p_j$ is the fraction of summation terms in stratum $j$, and $q_j$ is the fraction of samples drawn from stratum $j$. Algorithm 3 modifies the **addSamples** subroutine to sample in stratified fashion, and computes the stratified $\hat{\mu}_{fs}$ and $\hat{\sigma}^2_{fs}$ instead of $\hat{\mu}_f$ and $\hat{\sigma}^2_f$ in **calcStats**. Since these estimators satisfy the two conditions necessary for the error guarantee, this establishes the theorem. □

The true variance $\sigma^2(\hat{\mu}_{fs})$ is identical to Equation 6 but with the exact $\sigma^2_j$ substituted for $\hat{\sigma}^2_j$. In [7], it is shown that $\sigma^2_{fs} \leq \sigma^2_f$, i.e. stratification never increases variance, and that any refinement of a

stratification can only reduce $\sigma^2_{fs}$. Although the sample allocation fractions $q_j$ can be chosen arbitrarily, $\sigma^2_{fs}$ is minimized when $q_j \propto p_j \sigma_j$. With this optimal allocation, $\sigma^2_{fs}$ reduces to $(\sum_j p_j \sigma_j)^2$. This motivates our $kd$-tree expansion heuristic, as described above, which tries to first split the nodes with highest $p_j \sigma_j$, i.e. the nodes with highest contribution to the variance under optimal allocation. While we never know the $\sigma_j$ exactly, Algorithm 3 uses the sample estimates $\hat{\sigma}_j$ at each stage to approximate the optimal allocation (this is the **optAlloc** routine).

Finally, the Theorem 2 sample complexity still holds for the CLT-governed stratified sample mean.

**Corollary 2.2.** *Given $m_{min}$ large enough to put $\hat{\mu}_{fs}$ and $\hat{\sigma}_{fs}$ in their asymptotic normal regimes, with probability at least $1 - 2\alpha$ Algorithm 3 terminates with $m \leq O\big(\frac{\sigma^2_{fs}}{\mu^2_f} + \frac{\sigma_{fs}}{|\mu_f|}\sqrt{\frac{\mu_{4fs}}{\sigma^4_{fs}} - 1}\big)$.*

**Correlated Sampling.** The variance of recursively estimated $f_i$, as expressed by $\nabla_f(\mathcal{G})\Sigma_m \nabla^T_f(\mathcal{G})$, depends on the full covariance matrix of the estimated arguments. If the gradient of $f$ is such that the variance of $f_i$ depends negatively (positively) on a covariance $\sigma_{jk}$, we can reduce the variance by inducing positive (negative) covariance between $G_j$ and $G_k$. Covariance can be induced by sharing sampled points across the estimates of $G_j$ and $G_k$, assuming they both use the same datasets. In some cases the expression for $f_i$'s variance is such that the effect of correlated sampling is data-dependent; when this happens, it is easy to test and check whether correlation helps. All experiments presented here were benefited by correlated sampling on top of stratification.

# 6  Experiments

We present experimental results in two phases. First, we compare stratified multi-stage Monte Carlo approximations to exact evaluations on tractable datasets. We show that the error distributions conform closely to our asymptotic theory. Second, having verified accuracy to the extent possible, we run our method on datasets containing millions of points in order to show 1) validation of the theoretical prediction that runtime is roughly independent of dataset size, and 2) many orders of magnitude speedup (as high as $10^{14}$) relative to exact computation. These results are presented for two method-dataset pairs: kernel regression on a dataset containing 2 million 4-dimensional redshift measurements used for quasar identification, and kernel conditional density estimation on an n-body galaxy simulation dataset containing 3.5 million 3-dimensional locations. In the KR case, the fourth dimension is regressed against the other three, while in KCDE the distribution of the third dimension is predicted as a function of the first two. In both cases we are evaluating the cross-validated score functions used for bandwidth optimization, i.e. $S_{KR}$ and $S_{KCDE}$ as described in Section 2.

**Error Control.** The objective of this first set of experiments is to validate the guarantee that relative error will be less than or equal to $\epsilon$ with probability $1 - \alpha$. We measured the distribution of error on a series of random data subsets up to the highest size for which the exact computation was tractable. For the $O(n^2)$ $S_{KR}$, the limit was $n = 10K$, while for the $O(n^3)$ $S_{KCDE}$ it was $n = 250$. For each dataset we randomly chose and evaluated 100 bandwidths with $1 - \alpha = 0.95$ and $\epsilon = 0.1$. Figure 1 shows the full quantile spreads of the relative errors. The most salient feature is the relationship of the 95% quantile line (dashed) to the threshold line at $\epsilon = 0.1$ (solid). Full compliance with asymptotic theory would require the dashed line never to be above the solid. This is basically the case for KCDE,[1] while the KR line never goes above 0.134. The approximation is therefore quite good, and could be improved if desired by increasing $m_{min}$ or the number of strata, but in this case we chose to trade a slight increase in error for an increase in speed.

**Speedup.** Given the validation of the error guarantees, we now turn to computational performance. As before, we ran on a series of random subsets of the data, this time with $n$ ranging into the millions. At each value of $n$, we randomly chose and evaluated 100 bandwidths, measuring the time for each evaluation. Figure 2 presents the average evaluation time versus dataset size for both methods. The most striking feature of these graphs is their flatness as $n$ increases by orders of magnitude. This is in accord with Theorem 2 and its corollaries, which predict sample and computational complexity independent of dataset size. Speedups[2] for KR range from 1.8 thousand at $n = 50K$ to 2.8 million at $n = 2M$. KCDE speedups range from 70 million at $n = 50K$ to $10^{14}$ at $n = 3.5M$. This performance is many orders of magnitude better than that of previous methods.

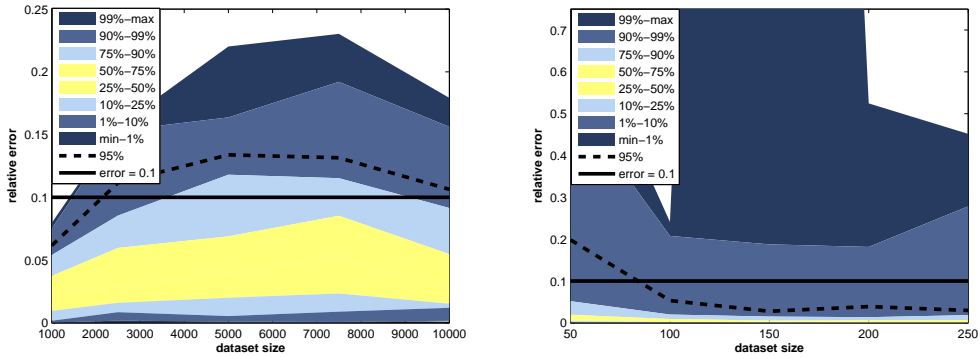

Figure 1: Error distribution vs. dataset size for KR (left), and KCDE (right).

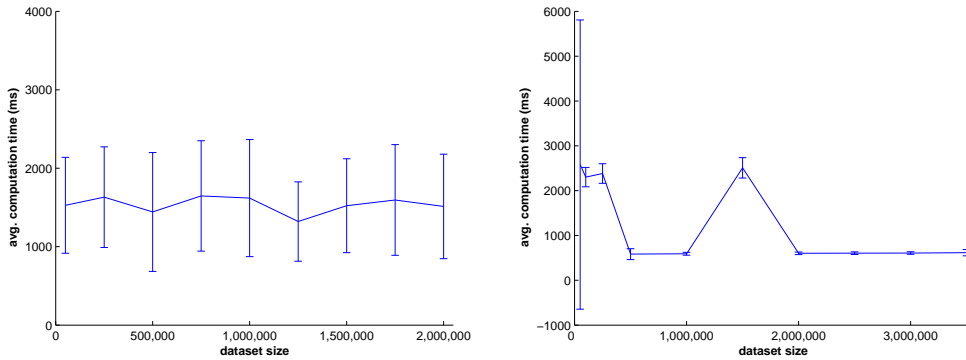

Figure 2: Runtime vs. dataset size for KR (left), and KCDE (right). Error bars are one standard deviation.

# 7 Conclusion

We have presented a multi-stage stratified Monte Carlo method for efficiently approximating a broad class of generalized nested summations. Summations of this type lead to computational bottlenecks in kernel estimators and elsewhere in machine learning. The theory derived for this Monte Carlo approach predicts: 1) relative error no greater than $\epsilon$ with probability at least $1-\alpha$, for user-specified $\epsilon$ and $\alpha$, and 2) sample and computational complexity independent of dataset size. Our experimental results validate these theoretical guarantees on real datasets, where we accelerate kernel cross-validation scores by as much as $10^{14}$ on millions of points. This is many orders of magnitude faster than the previous state of the art. In addition to applications, future work will likely include automatic selection of stratification granularity, additional variance reduction techniques, and further generalization to other computational bottlenecks such as linear algebraic operations.

## Footnotes

[1]The spike in the max quantile is due to a single outlier point.

[2]All speedups are relative to extrapolated runtimes based on the $O()$ order of the exact computation.

# References

[1] Nicol N. Schraudolph and Thore Graepel. Combining conjugate direction methods with stochastic approximation of gradients. In *Workshop on Artificial Intelligence and Statistics (AISTATS)*, 2003.

[2] Alexander G. Gray and Andrew W. Moore. N-body problems in statistical learning. In *Advances in Neural Information Processing Systems (NIPS) 13*, 2000.

[3] Mike Klaas, Mark Briers, Nando de Freitas, and Arnaud Doucet. Fast particle smoothing: If I had a million particles. In *International Conference on Machine Learning (ICML)*, 2006.

[4] Ping Wang, Dongryeol Lee, Alexander Gray, and James M. Rehg. Fast mean shift with accurate and stable convergence. In *Workshop on Artificial Intelligence and Statistics (AISTATS)*, 2007.

[5] Michael P. Holmes, Alexander G. Gray, and Charles Lee Isbell Jr. Fast nonparametric conditional density estimation. In *Uncertainty in Artificial Intelligence (UAI)*, 2007.

[6] Reuven Y. Rubinstein. *Simulation and the Monte Carlo Method.* John Wiley & Sons, 1981.

[7] Paul Glasserman. *Monte Carlo methods in financial engineering.* Springer-Verlag, 2004.

